# Tensor Decomposition for Fast Parsing with Latent-Variable PCFGs

**Shay B. Cohen and Michael Collins**
Department of Computer Science
Columbia University
New York, NY 10027
scohen,mcollins@cs.columbia.edu

## Abstract

We describe an approach to speed-up inference with latent-variable PCFGs, which have been shown to be highly effective for natural language parsing. Our approach is based on a tensor formulation recently introduced for spectral estimation of latent-variable PCFGs coupled with a tensor decomposition algorithm well-known in the multilinear algebra literature. We also describe an error bound for this approximation, which gives guarantees showing that if the underlying tensors are well approximated, then the probability distribution over trees will also be well approximated. Empirical evaluation on real-world natural language parsing data demonstrates a significant speed-up at minimal cost for parsing performance.

## 1 Introduction

Latent variable models have shown great success in various fields, including computational linguistics and machine learning. In computational linguistics, for example, latent-variable models are widely used for *natural language parsing* using models called latent-variable PCFGs (L-PCFGs; [14]).

The mainstay for estimation of L-PCFGs has been the expectation-maximization algorithm [14, 16], though other algorithms, such as spectral algorithms, have been devised [5]. A by-product of the spectral algorithm presented in [5] is a *tensor formulation* for computing the inside-outside probabilities of a L-PCFG. Tensor products (or matrix-vector products, in certain cases) are used as the basic operation for *marginalization* over the latent annotations of the L-PCFG.

The computational complexity with the tensor formulation (or with plain CKY, for that matter) is cubic in the number of latent states in the L-PCFG. This multiplicative factor can be prohibitive for a large number of hidden states; various heuristics are used in practice to avoid this problem [16].

In this paper, we show that *tensor decomposition* can be used to significantly speed-up the parsing performance with L-PCFGs. Our approach is also provided with a theoretical guarantee: given the accuracy of the tensor decomposition, one can compute how accurate the approximate parser is.

The rest of this paper is organized as follows. We give notation and background in §2–3, and then present the main approach in §4. We describe experimental results in §5 and conclude in §6.

## 2 Notation

Given a matrix $A$ or a vector $v$, we write $A^\top$ or $v^\top$ for the associated transpose. For any integer $n \geq 1$, we use $[n]$ to denote the set $\{1, 2, \ldots n\}$. We will make use of tensors of rank 3:[1]

**Definition 1.** *A tensor $C \in \mathbb{R}^{(m \times m \times m)}$ is a set of $m^3$ parameters $C_{i,j,k}$ for $i, j, k \in [m]$. Given a tensor $C$, and vectors $y^1 \in \mathbb{R}^m$ and $y^2 \in \mathbb{R}^m$, we define $C(y^1, y^2)$ to be the $m$-dimensional row vector with components $[C(y^1, y^2)]_i = \sum_{j \in [m], k \in [m]} C_{i,j,k} y_j^1 y_k^2$. Hence $C$ can be interpreted as a function $C : \mathbb{R}^m \times \mathbb{R}^m \to \mathbb{R}^{1 \times m}$ that maps vectors $y^1$ and $y^2$ to a row vector $C(y^1, y^2) \in \mathbb{R}^{1 \times m}$.*

*In addition, we define the tensor $C_{(1,2)} \in \mathbb{R}^{(m \times m \times m)}$ for any tensor $C \in \mathbb{R}^{(m \times m \times m)}$ to be the function $C_{(1,2)} : \mathbb{R}^m \times \mathbb{R}^m \to \mathbb{R}^{m \times 1}$ defined as $[C_{(1,2)}(y^1, y^2)]_k = \sum_{i \in [m], j \in [m]} C_{i,j,k} y_i^1 y_j^2$. Similarly, for any tensor $C$ we define $C_{(1,3)} : \mathbb{R}^m \times \mathbb{R}^m \to \mathbb{R}^{m \times 1}$ as $[C_{(1,3)}(y^1, y^2)]_j = \sum_{i \in [m], k \in [m]} C_{i,j,k} y_i^1 y_k^2$. Note that $C_{(1,2)}(y^1, y^2)$ and $C_{(1,3)}(y^1, y^2)$ are both **column** vectors.*

For two vectors $x \in \mathbb{R}^m$ and $y \in \mathbb{R}^m$ we denote by $x \odot y \in \mathbb{R}^m$ the Hadamard product of $x$ and $y$, i.e. $[x \odot y]_i = x_i y_i$. Finally, for vectors $x, y, z \in \mathbb{R}^m$, $xy^\top z^\top$ is the tensor $D \in \mathbb{R}^{m \times m \times m}$ where $D_{i,j,k} = x_i y_j z_k$ (this is analogous to the outer product: $[xy^\top]_{i,j} = x_i y_j$).

## 3 Latent-Variable Parsing

In this section we describe latent-variable PCFGs and their parsing algorithms.

### 3.1 Latent-Variable PCFGs

This section gives a definition of the L-PCFG formalism used in this paper; we follow the definitions given in [5]. An L-PCFG is a 5-tuple $(\mathcal{N}, \mathcal{I}, \mathcal{P}, m, n)$ where:

- $\mathcal{N}$ is the set of non-terminal symbols in the grammar. $\mathcal{I} \subset \mathcal{N}$ is a finite set of *in-terminals*. $\mathcal{P} \subset \mathcal{N}$ is a finite set of *pre-terminals*. We assume that $\mathcal{N} = \mathcal{I} \cup \mathcal{P}$, and $\mathcal{I} \cap \mathcal{P} = \emptyset$. Hence we have partitioned the set of non-terminals into two subsets.
- $[m]$ is the set of possible hidden states.
- $[n]$ is the set of possible words.
- For all $a \in \mathcal{I}, b \in \mathcal{N}, c \in \mathcal{N}, h_1, h_2, h_3 \in [m]$, we have a context-free rule $a(h_1) \to b(h_2) \ c(h_3)$.
- For all $a \in \mathcal{P}, h \in [m], x \in [n]$, we have a context-free rule $a(h) \to x$.

Note that for any binary rule, $a \to b \ c$, it holds that $a \in \mathcal{I}$, and for any unary rule $a \to x$, it holds that $a \in \mathcal{P}$.

The set of "skeletal rules" is defined as $\mathcal{R} = \{a \to b \ c : a \in \mathcal{I}, b \in \mathcal{N}, c \in \mathcal{N}\}$. The parameters of the model are as follows:

- For each $a \to b \ c \in \mathcal{R}$, and $h_1, h_2, h_3 \in [m]$, we have a parameter $t(a \to b \ c, h_2, h_3 | h_1, a)$.
- For each $a \in \mathcal{P}, x \in [n]$, and $h \in [m]$, we have a parameter $q(a \to x | h, a)$.

An L-PCFG corresponds to a regular PCFG with non-terminals annotated with latent states. For each triplet of latent states and a rule $a \to b \ c$, we have a rule probability $p(a(h_1) \to b(h_2) \ c(h_3) | a(h_1)) = t(a \to b \ c, h_2, h_3 | h_1, a)$. Similarly, we also have parameters $p(a(h) \to x | a(h)) = q(a \to x | h, a)$. In addition, there are initial probabilities of generating a non-terminal with a latent at the top of the tree, denoted by $\pi(a, h)$.

L-PCFGs induce distributions over two type of trees: *skeletal trees*, i.e. trees without values for latent states (these trees are observed in data), and *full trees* (trees with values for latent states). A skeletal tree consists of a sequence of rules $r_1 \dots r_N$ where $r_i \in \mathcal{R}$ or $r_i = a \to x$. See Figure 3.1 for an example.

We now turn to the problem of computing the probability of a skeletal tree, by marginalizing out the latent states of full trees. Let $r_1 \dots r_N$ be a derivation, and let $a_i$ be the non-terminal on the left hand-side of rule $r_i$. For any $r_i = a \to b \ c$, define $h_i^{(2)}$ to be the latent state associated with the left child of the rule $r_i$ and $h_i^{(3)}$ to be the hidden variable value associated with the right child.

The distribution over full trees is then:

$$p(r_1 \dots r_N, h_1 \dots h_N) = \pi(a_1, h_1) \times \prod_{i:a_i \in \mathcal{I}} t(r_i, h_i^{(2)}, h_i^{(3)} | h_i, a_i) \times \prod_{i:a_i \in \mathcal{P}} q(r_i | h_i, a_i)$$

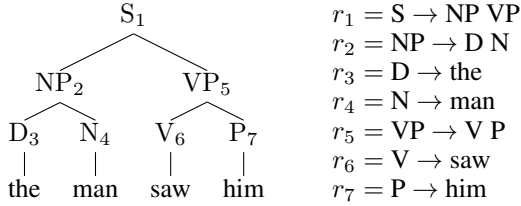

$r_1 = S \to NP\ VP$
$r_2 = NP \to D\ N$
$r_3 = D \to the$
$r_4 = N \to man$
$r_5 = VP \to V\ P$
$r_6 = V \to saw$
$r_7 = P \to him$

Figure 1: An s-tree with its sequence of rules. (The nodes in the tree are indexed by the derivation order, which is canonicalized as top-down, left-most derviation.)

Marginalizing out the latent states leads to the distribution over the skeletal tree $r_1 \ldots r_N$:
$p(r_1 \ldots r_N) = \sum_{h_1 \ldots h_N} p(r_1 \ldots r_N, h_1 \ldots h_N)$.

It will be important for the rest of this paper to use of matrix form of parameters of an L-PCFG, as follows:

- For each $a \to b\ c \in \mathcal{R}$, we define $T^{a \to b\ c} \in \mathbb{R}^{m \times m \times m}$ to be the tensor with values

$$T^{a \to b\ c}_{h_1, h_2, h_3} = t(a \to b\ c, h_2, h_3 | a, h_1)$$

- For each $a \in \mathcal{P}$, $x \in [n]$, we define $Q^{a \to x} \in \mathbb{R}^{1 \times m}$ to be the vector with values $q(a \to x | h, a)$ for $h = 1, 2, \ldots m$.
- For each $a \in \mathcal{I}$, we define the vector $\pi^a \in \mathbb{R}^m$ where $[\pi^a]_h = \pi(a, h)$.

**Parameter Estimation**    Several ways to estimate the parameters $T^{a \to b\ c}$, $Q^{a \to x}$ and $\pi^a$ have been suggested in the literature. For example, vanilla EM has been used in [14], hierarchical state splitting EM has been suggested in [16], and a spectral algorithm is proposed in [5].

In the rest of the paper, we assume that the parameters for these tensors have been identified, and focus mostly on the problem of *inference* – i.e. parsing unseen sentences. The reason for this is two-fold: (a) in real-world applications, training can be done off-line to identify a set of parameters once, and therefore its computational efficiency is of lesser interest; (b) our approach can speed-up the inference problems existing in the EM algorithm, but the speed-up is of lesser interest, because the inference problem in the EM algorithm is linear in the tree size (and not cubic, as in the case of parsing). The reason for this linear complexity is that the skeletal trees are observed during EM training. Still, EM stays cubic in the number of states.

### 3.2   Tensor Formulation for Inside-Outside

There are several ways to parse a sentence with latent-variable PCFGs. Most of these approaches are taken by using an *inside-outside* algorithm [12] which computes *marginals* for various non-terminals and spans in the sentence, and then eventually finding a parse tree which maximizes a score which is the sum of the marginals of the spans that appear in the tree.

More formally, let $\mu(a, i, j) = \sum_{\tau \in \mathcal{T}(x):(a,i,j) \in \tau} p(\tau)$ for each non-terminal $a \in \mathcal{N}$, for each $(i, j)$ such that $1 \leq i \leq j \leq N$. Here $\mathcal{T}(x)$ denotes the set of all possible s-trees for the sentence $x$, and we write $(a, i, j) \in \tau$ if non-terminal $a$ spans words $x_i \ldots x_j$ in the parse tree $\tau$.

Then, the parsing algorithm seeks for a given sentence $x = x_1 \ldots x_N$ the skeletal tree $\arg\max_{\tau \in \mathcal{T}(x)} \sum_{(a,i,j) \in \tau} \mu(a, i, j)$.

Given the marginals $\mu(a, i, j)$, one can use the dynamic programming algorithm described in [7] in order to find this highest scoring tree.

A key question is how to compute the marginals $\mu(a, i, j)$ using the inside-outside algorithm. Dynamic programming solutions are available for this end as well. The complexity of a naïve implementation of the dynamic programming algorithm for this problem is cubic in the number of latent states. This is where we suggest an alternative to the traditional dynamic programming solutions. Our alternative relies on an existing tensor formulation for the inside-outside algorithm [5], which re-formalizes the dynamic programming algorithm using tensor, matrix and vector product operations.

Algorithm 2 presents the re-formulation of the inside-outside algorithm using tensors. For more details and proofs of correctness, refer to [5]. The re-formalized algorithm is still cubic in

the number of hidden states, and spends most of the time computing the tensor applications $T^{a \to b\ c}(\alpha^{b,i,k}, \alpha^{c,k+1,j})$, $T_{(1,2)}^{b \to c\ a}(\beta^{b,k,j}, \alpha^{c,k,i-1})$ and $T_{(1,3)}^{b \to a\ c}(\beta^{b,i,k}, \alpha^{c,j+1,k})$. This is the main set of computations we aim to speed-up, as we show in the next section.

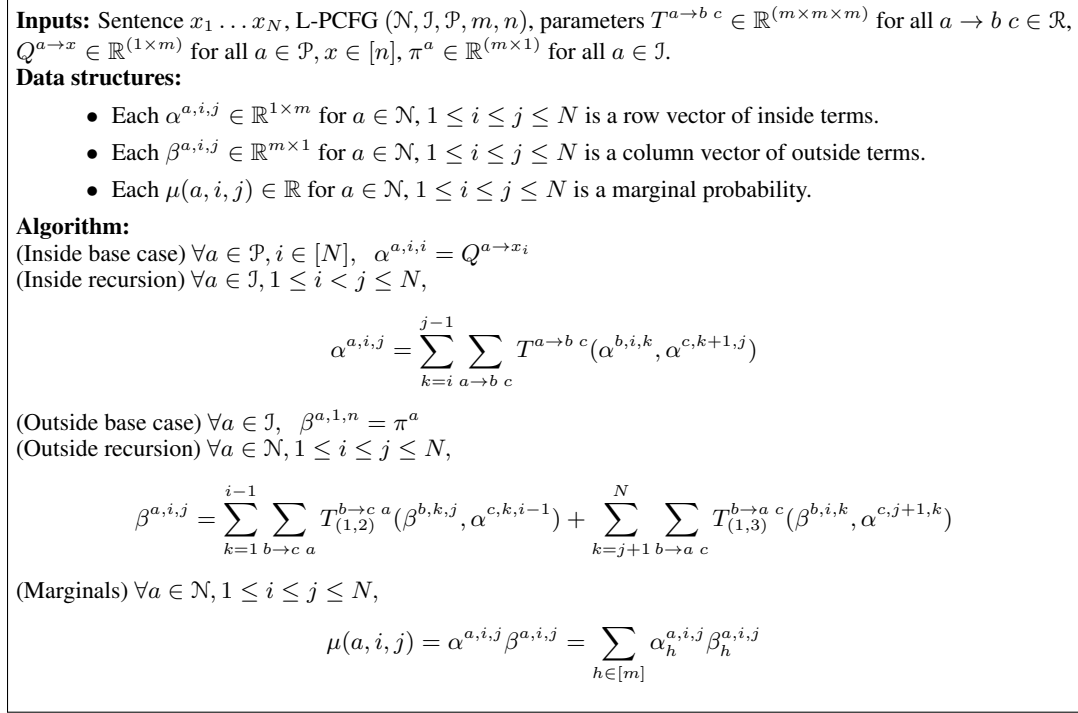

**Inputs:** Sentence $x_1 \dots x_N$, L-PCFG $(\mathcal{N}, \mathcal{I}, \mathcal{P}, m, n)$, parameters $T^{a \to b\ c} \in \mathbb{R}^{(m \times m \times m)}$ for all $a \to b\ c \in \mathcal{R}$, $Q^{a \to x} \in \mathbb{R}^{(1 \times m)}$ for all $a \in \mathcal{P}, x \in [n]$, $\pi^a \in \mathbb{R}^{(m \times 1)}$ for all $a \in \mathcal{I}$.
**Data structures:**

- Each $\alpha^{a,i,j} \in \mathbb{R}^{1 \times m}$ for $a \in \mathcal{N}, 1 \le i \le j \le N$ is a row vector of inside terms.
- Each $\beta^{a,i,j} \in \mathbb{R}^{m \times 1}$ for $a \in \mathcal{N}, 1 \le i \le j \le N$ is a column vector of outside terms.
- Each $\mu(a, i, j) \in \mathbb{R}$ for $a \in \mathcal{N}, 1 \le i \le j \le N$ is a marginal probability.

**Algorithm:**
(Inside base case) $\forall a \in \mathcal{P}, i \in [N], \ \alpha^{a,i,i} = Q^{a \to x_i}$
(Inside recursion) $\forall a \in \mathcal{I}, 1 \le i < j \le N$,

$$\alpha^{a,i,j} = \sum_{k=i}^{j-1} \sum_{a \to b\ c} T^{a \to b\ c}(\alpha^{b,i,k}, \alpha^{c,k+1,j})$$

(Outside base case) $\forall a \in \mathcal{I}, \ \beta^{a,1,n} = \pi^a$
(Outside recursion) $\forall a \in \mathcal{N}, 1 \le i \le j \le N$,

$$\beta^{a,i,j} = \sum_{k=1}^{i-1} \sum_{b \to c\ a} T_{(1,2)}^{b \to c\ a}(\beta^{b,k,j}, \alpha^{c,k,i-1}) + \sum_{k=j+1}^{N} \sum_{b \to a\ c} T_{(1,3)}^{b \to a\ c}(\beta^{b,i,k}, \alpha^{c,j+1,k})$$

(Marginals) $\forall a \in \mathcal{N}, 1 \le i \le j \le N$,

$$\mu(a, i, j) = \alpha^{a,i,j} \beta^{a,i,j} = \sum_{h \in [m]} \alpha_h^{a,i,j} \beta_h^{a,i,j}$$

Figure 2: The tensor form of the inside-outside algorithm, for calculation of marginal terms $\mu(a, i, j)$.

## 4 Tensor Decomposition

As mentioned earlier, most computation for the inside-outside algorithm is spent on the tensor calculation of $T^{a \to b\ c}$ on the intermediate inside/outside quantities. These computations, appearing as $T^{a \to b\ c}(\alpha^{b,i,k}, \alpha^{c,k+1,j})$, $T_{(1,2)}^{b \to c\ a}(\beta^{b,k,j}, \alpha^{c,k,i-1})$ and $T_{(1,3)}^{b \to a\ c}(\beta^{b,i,k}, \alpha^{c,j+1,k})$ output a vector of length $m$, where computation of each element in the vector is $\mathcal{O}(m^2)$. Therefore, the inside-outside has a multiplicative $\mathcal{O}(m^3)$ factor in its computational complexity, which we aim to reduce.

For the rest of this section, fix a binary grammar rule $a \to b\ c$ and consider the tensor $T \triangleq T^{a \to b\ c}$ associated with it. Consider a pair of two vectors $y^1, y^2 \in \mathbb{R}^m$, associated with the distributions over latent-states for the left ($y^1$) and right child ($y^2$) of a given node in a parse tree. Our method for improving the speed of this tensor computation relies on a simple observation. Given an integer $r \ge 1$, assume that the tensor $T$ had the following special form, which is also called "Kruskal form", $T = \sum_{i=1}^{r} u_i v_i^\top w_i^\top$, i.e. it would be the sum of $r$ tensors, each is the tensor product of three vectors. In that case, the cost of computing $T(y^1, y^2)$ could be greatly reduced by computing:

$$T(y^1, y^2) = \left[ \sum_{i=1}^{r} u_i v_i^\top w_i^\top \right] (y^1, y^2) = \sum_{i=1}^{r} u_i (v_i^\top y^1)(w_i^\top y^2) = U^\top (V y^1 \odot W y^2) \qquad (1)$$

where $U, V, W \in \mathbb{R}^{r \times m}$ with the $i$th row being $u_i$, $v_i$ and $w_i$ respectively.

The total complexity of this computation is $\mathcal{O}(mr)$. We see later that our approach can be used effectively for $r$ as small as 2, turning the inside-outside algorithm for latent-variable PCFGs into a linear algorithm in the number of hidden states.

We note that it is well-known that an exact tensor decomposition can be achieved by using $r = m^2$ [11]. In that case, there is no computational gain. The minimal $r$ required for an exact solution can be smaller than $m^2$, but identifying that minimal $r$ is NP-hard [9].

We focused on this section on the computation $T^{a \to b\ c}(\alpha^{b,i,k}, \alpha^{c,k+1,j})$, but the steps above can be generalized easily for the cases of computing $T^{b \to c\ a}_{(1,2)}(\beta^{b,k,j}, \alpha^{c,k,i-1})$ and $T^{b \to a\ c}_{(1,3)}(\beta^{b,i,k}, \alpha^{c,j+1,k})$.

## 4.1 CP Decomposition of Tensors

In the general case, for a fixed $r$, our latent-variable PCFG tensors will not have the exact decomposed form from the previous section. Still, by using decomposition algorithms from multilinear algebra, we can *approximate* the latent-variable tensors, where the quality of approximation is measured according to some norm over the set of tensors $\mathbb{R}^{m \times m \times m}$.

An example of such a decomposition is the canonical polyadic decomposition (CPD), also known as CANDECOMP/PARAFAC decomposition [3, 8, 10]. Given an integer $r$, *least squares* CPD aims to find the nearest tensor in Kruskal form according to the analogous norm (for tensors) to the Frobenius norm (for matrices).

More formally, for a given tensor $D \in \mathbb{R}^{m \times m \times m}$, let $||D||_F = \sqrt{\sum_{i,j,k} D^2_{i,j,k}}$. Let the set of tensors in Kruskal form $\mathcal{C}_r$ be:

$$\mathcal{C}_r = \{C \in \mathbb{R}^{m \times m \times m} \mid C = \sum_{i=1}^{r} u_i v_i^\top w_i^\top \ s.t. \ u_i, v_i, w_i \in \mathbb{R}^m\}.$$

The least squares CPD of $C$ is a tensor $\hat{C}$ such that $\hat{C} \in \arg\min_{\hat{C} \in \mathcal{C}_r} ||C - \hat{C}||_F$.

There are various algorithms to perform CPD, such as alternating least squares, direct linear decomposition, alternating trilinear decomposition and pseudo alternating least squares [6]. Most of these implementations treat the problem of identifying the approximate tensor as an optimization problem. These algorithms are not exact. Any of these implementations can be used in our approach. We note that the decomposition optimization problem is hard, and often has multiple local maxima. Therefore, the algorithms mentioned above are inexact.

In our experiments, we use the alternating least squares algorithm. This method works by iteratively improving $U$, $V$ and $W$ from Eq. 1 (until convergence), each time solving a least squares problem.

## 4.2 Propagation of Errors

We next present a theoretical guarantee about the quality of the CP-approximated tensor formulation of the inside-outside algorithm. We measure the propagation of errors in probability calculations through a given parse tree. We derive a similar result for the marginals.

We denote by $\hat{p}$ the distribution induced over trees (skeletal and full), where we approximate each $T^{a \to b\ c}$ using the tensor $\hat{T}^{a \to b\ c}$. Similarly, we denote by $\hat{\mu}(a, i, j)$ the approximated marginals.

**Lemma 4.1.** *Let $C \in \mathbb{R}^{m \times m \times m}$ and let $y^1, y^2, \hat{y}^1, \hat{y}^2 \in \mathbb{R}^m$. Then the following inequalities hold:*

$$||C(y^1, y^2)||_2 \leq ||C||_F ||y^1||_2 ||y^2||_2 \tag{2}$$

$$||C(y^1, y^2) - C(\hat{y}^1, \hat{y}^2)||_2 \leq ||C||_F \max\{||y^1||_2, ||\hat{y}^2||_2\}(||y^1 - \hat{y}^1||_2 + ||y^2 - \hat{y}^2||_2) \tag{3}$$

*Proof.* Eq. 2 is the result of applying Cauchy-Schwarz inequality twice:

$$||C(y^1, y^2)||_2^2 = \sum_i \left(\sum_{j,k} C_{i,j,k} y_j^1 y_k^2\right)^2 \leq \sum_i \left(\sum_{j,k} C_{i,j,k}^2\right)\left(\sum_j (y_j^1)^2\right)\left(\sum_k (y_k^2)^2\right)$$

$$= ||C||_F^2 \cdot ||y^1||_2^2 \cdot ||y^2||_2^2$$

For Eq. 3, note that $C(y^1, y^2) - C(\hat{y}^1, \hat{y}^2) = C(y^1, y^2) - C(y^1, \hat{y}^2) + C(y^1, \hat{y}^2) - C(\hat{y}^1, \hat{y}^2)$, and therefore from the triangle inequality and bi-linearity of $C$:

$$||C(y^1, y^2) - C(\hat{y}^1, \hat{y}^2)||_2 \leq ||C(y^1, y^2 - \hat{y}^2)||_2 + ||C(y^1 - \hat{y}^1, \hat{y}^2)||_2$$
$$\leq ||C||_F \left( ||y^1||_2 ||y^2 - \hat{y}^2||_2 + ||y^1 - \hat{y}^1||_2 ||\hat{y}^2||_2 \right)$$
$$\leq ||C||_F \max\{||y^1||_2, ||\hat{y}^2||_2\}(||y^1 - \hat{y}^1||_2 + ||y^2 - \hat{y}^2||_2)$$

$\square$

Equipped with this Cauchy-Schwarz style lemma, we can prove the following theorem:

**Theorem 4.2.** *Let* $d^* = \dfrac{\log(\frac{1}{\gamma}) + 1}{\log(2(\sqrt{m} + 1)) + \log(\gamma + \sqrt{m})}$ *where* $\gamma$ *is the the "tensor approximation error" defined as* $\gamma = \max_{a \to b\ c} ||T^{a \to b\ c} - \hat{T}^{a \to b\ c}||_F$, *then:*

- *For a given skeletal tree* $r_1, \dots, r_N$, *if the depth of the tree, denoted* $d$, *is such that*

$$d \leq \min \left\{ \frac{\log(\frac{1}{\gamma}) - \log(\frac{m}{\epsilon})}{\log(2(\sqrt{m} + 1)) + \log(\gamma + \sqrt{m})}, d^* \right\} \quad \text{then } |p(r_1, \dots, r_N) - \hat{p}(r_1, \dots, r_N)| \leq \epsilon$$

- *For a given sentence* $x_1, \dots, x_M$, *it holds that for all triplets* $(a, i, j)$, *if*

$$M \leq \min \left\{ \frac{\log(\frac{1}{\gamma}) - \log(\frac{m}{\epsilon})}{2\log(4|\mathcal{N}|) + \log(2(\sqrt{m} + 1)) + \log(\gamma + \sqrt{m})}, d^* \right\} \quad \text{then } |\mu(a, i, j) - \hat{\mu}(a, i, j)| \leq \epsilon$$

*Proof.* For the first part, the proof is using structural induction on the structure of the test tree. Assume a fixed skeletal tree $r_1, \dots, r_N$. The probability $p(r_1, \dots, r_N)$ can be computed by using a sequence of applications of $T^{a \to b\ c}$ on distribution over latent states for left and right children. More specifically, it can be shown that the vector of probabilities defined as $[y^i]_h = p(t_i \mid a_i, h_i = h)$ (ranging over $[m]$), where $t_i$ is the skeletal subtree rooted at node $i$ can be defined recursively as:

- $y^i = Q^{a \to x_i}$ if $i$ is a leaf node with word $x_i$ and,
- $y^i = T^{a \to b\ c}(y^j, y^k)$ if $i$ is a non-leaf node with node $j$ being the left child and node $k$ being the right child of node $i$.

Define the same quantities $\hat{y}^i$, only using the approximate tensors $\hat{T}^{a \to b\ c}$. Let $\delta_i = ||y^i - \hat{y}^i||$. We will prove inductively that if $d_i$ is the depth of the subtree at node $i$, then:

$$\delta_i \leq \min \left\{ \gamma m \left( \frac{(2(\sqrt{m} + 1)(\gamma + \sqrt{m}))^{d_i} - 1}{2(\sqrt{m} + 1)(\gamma + \sqrt{m}) - 1} \right), 1 \right\}$$

For any leaf node (base case): $||y^i - \hat{y}^i||_2 = 0$. For a given non-leaf node $i$:

$$\delta_i = ||y^i - \hat{y}^i||_2 = ||T^{a \to b\ c}(y^j, y^k) - \hat{T}^{a \to b\ c}(\hat{y}^j, \hat{y}^k)||_2$$
$$\leq ||T^{a \to b\ c}(y^j, y^k) - \hat{T}^{a \to b\ c}(y^j, y^k)||_2 + ||\hat{T}^{a \to b\ c}(y^j, y^k) - \hat{T}^{a \to b\ c}(\hat{y}^j, \hat{y}^k)||_2 \quad (4)$$
$$\leq ||T^{a \to b\ c} - \hat{T}^{a \to b\ c}||_F ||y^j||_2 ||y^k||_2$$
$$\quad + ||\hat{T}^{a \to b\ c}|| \max\{||y^j||_2, ||\hat{y}^k||_2\}(||y^j - \hat{y}^j||_2 + ||y^k - \hat{y}^k||_2) \quad (5)$$
$$\leq \gamma m + (\sqrt{m} + 1)(\gamma + \sqrt{m})(\delta_j + \delta_k)$$
$$\leq \gamma m \left( 1 + 2(\sqrt{m} + 1)(\gamma + \sqrt{m}) \frac{(2(\sqrt{m} + 1)(\gamma + \sqrt{m}))^{d_i - 1} - 1}{2(\sqrt{m} + 1)(\gamma + \sqrt{m}) - 1} \right) \quad (6)$$
$$= \gamma m \left( \frac{(2(\sqrt{m} + 1)(\gamma + \sqrt{m}))^{d_i} - 1}{2(\sqrt{m} + 1)(\gamma + \sqrt{m}) - 1} \right)$$

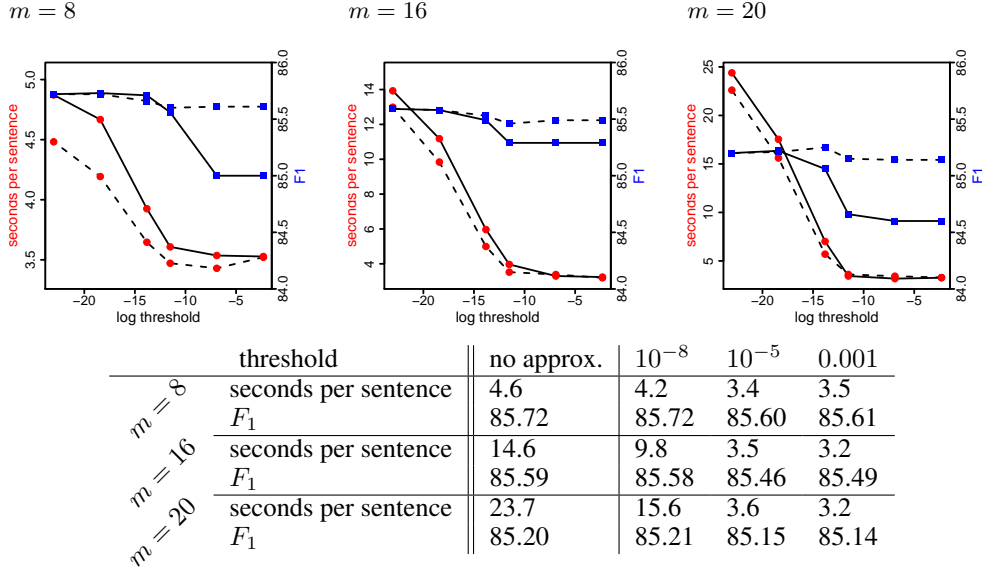

| threshold | | no approx. | $10^{-8}$ | $10^{-5}$ | 0.001 |
|---|---|---|---|---|---|
| $m=8$ | seconds per sentence | 4.6 | 4.2 | 3.4 | 3.5 |
| | $F_1$ | 85.72 | 85.72 | 85.60 | 85.61 |
| $m=16$ | seconds per sentence | 14.6 | 9.8 | 3.5 | 3.2 |
| | $F_1$ | 85.59 | 85.58 | 85.46 | 85.49 |
| $m=20$ | seconds per sentence | 23.7 | 15.6 | 3.6 | 3.2 |
| | $F_1$ | 85.20 | 85.21 | 85.15 | 85.14 |

Figure 3: Speed and performance of parsing with tensor decomposition for $m \in \{8, 16, 20\}$ (left plots, middle plots and right plots respectively). The left y axis is running time (red circles), the right y axis is $F_1$ performance of the parser (blue squares), the x axis corresponds to $\log t$. Solid lines describe decomposition with $r = 2$, dashed lines describe decomposition with $r = 8$. In addition, we include the numerical results for various $m$ for $r = 8$.

where Eq. 4 is the result of the triangle inequality, Eq. 5 comes from Lemma 4.1 and the fact that $||\hat{T}^{a \to b\ c}||_F \leq ||\hat{T}^{a \to b\ c} - T^{a \to b\ c}||_F + ||T^{a \to b\ c}||_F \leq \gamma + \sqrt{m}$ and $||\hat{y}^k||_2 \leq \delta_k + ||y^k||_2 \leq 1 + \sqrt{m}$ for any node $k$ (under ind. hyp.), and Eq. 6 is the result of applying the induction hypothesis. It can also be verified that since $d_i \leq d \leq d^*$ we have $\delta_i \leq 1$.

Since $m \geq 1$, it holds that $\delta_i \leq \gamma m \left(2(\sqrt{m} + 1)(\gamma + \sqrt{m})\right)^{d_i}$. Consider $|p(r_1, \ldots, r_N) - \hat{p}(r_1, \ldots, r_N)| = |\pi^{a_1}(y^1 - \hat{y}^1)| \leq ||\pi^{a_1}||_2 \delta_1 \leq \delta_1$ where $a_1$ is the non-terminal at the root of the tree. It is easy to verify that if $d_1 \leq \dfrac{\log(\frac{1}{\gamma}) - \log(\frac{m}{\epsilon})}{\log(2(\sqrt{m} + 1)) + \log(\gamma + \sqrt{m})}$, then $\delta_1 \leq \epsilon$, as needed.

For the marginals, consider that: $|\mu(a, i, j) - \hat{\mu}(a, i, j)| \leq \sum_{\tau \in \mathcal{T}(x)} |p(\tau) - \hat{p}(\tau)|$.

We have $d_1 \leq M$. In addition, if

$$M \leq \frac{\log(\frac{1}{\gamma}) - 2M \log(4|\mathcal{N}|) - \log(\frac{m}{\epsilon})}{\log(2(\sqrt{m} + 1)) + \log(\gamma + \sqrt{m})} \text{ then } d_1 \leq \frac{\log(\frac{1}{\gamma}) - \log(\frac{m}{\epsilon/|\mathcal{T}(x)|})}{\log(2(\sqrt{m} + 1)) + \log(\gamma + \sqrt{m})} \quad (7)$$

because the number of labeled binary trees for a sentence of length $M$ is at most $(4|\mathcal{N}|)^{2M}$ (and therefore $|\mathcal{T}(x)| \leq (4|\mathcal{N}|)^{2M}$; $4^l$ is a bound on the Catalan number, the number of binary trees over $l$ nodes), then $|\mu(a, i, j) - \hat{\mu}(a, i, j)| \leq \epsilon$.

It can be verified that the left hand-side of Eq. 7 is satisfied if $M \leq \dfrac{\log(\frac{1}{\gamma}) - \log(\frac{m}{\epsilon})}{2 \log(4|\mathcal{N}|) + \log(2(\sqrt{m} + 1)) + \log(\gamma + \sqrt{m})}$. $\quad \square$

As expected, the longer a sentence is, or the deeper a parse tree is, the better we need the tensor approximation to be (smaller $\gamma$) for the inside-outside to be more accurate.

## 5 Experiments

We devote this section to empirical evaluation of our approach. Our goal is to evaluate the trade-off between the accuracy of the tensor decomposition and the speed-up in the parsing algorithm.

**Experimental Setup**  We use sections 2–21 of the Penn treebank [13] to train a latent-variable parsing model using the expectation-maximization algorithm (EM was run for 15 iterations) for various number of latent states ($m \in \{8, 16, 20\}$), and then parse in various settings section 22 of the same treebank (sentences of length $\leq 40$). Whenever we report parsing accuracy, we use the traditional $F_1$ measure from the Parseval metric [2]. It computes the $F_1$ measure of spans $(a, i, j)$ appearing in the gold standard and the hypothesized trees.

The total number of tensors extracted from the training data using EM was 7,236 (corresponding to the number of grammar rules). Let $\gamma_{a \to b\,c} = ||T^{a \to b\,c} - \hat{T}^{a \to b\,c}||_F$. In our experiments, we vary a threshold $t \in \{0.1, 0.001, 10^{-5}, 10^{-6}, 10^{-8}, 0\}$ – an approximate tensor $\hat{T}^{a \to b\,c}$ is used instead of $T^{a \to b\,c}$ only if $\gamma_{a \to b\,c} \leq t$. The value $t = 0$ implies using vanilla inference, without any approximate tensors. We describe experiments with $r \in \{2, 8\}$. For the tensor approximation, we use the implementation provided in the Matlab tensor toolbox from [1]. The toolbox implements the alternating least squares method.

As is common, we use a pruning technique to make the parser faster – items in the dynamic programming chart are pruned if their value according to a base vanilla maximum likelihood model is less than 0.00005 [4]. We report running times considering this pruning as part of the execution. The parser was run on a single Intel Xeon 2.67GHz CPU.

We note that the performance of the parser improves as we add more latent states. The performance of the parser with vanilla PCFG ($m = 1$) is 70.26 $F_1$ measure.

**Experimental Results**  Table 3 describes $F_1$ performance and running time as we vary $t$. It is interesting to note that the speed-up, for the same threshold $t$, seems to be larger when using $r = 8$ instead of $r = 2$. At first this may sound counter-intuitive. The reason for this happening is that with $r = 8$, more of the tensors have an approximation error which is smaller than $t$, and therefore more approximate tensors are used than in the case of $r = 2$.

Using $t = 0.1$, the speed-up is significant over non-approximate version of the parser. More specifically, for $r = 8$, it takes 72% of the time (without considering the pruning phase) of the non-approximate parser to parse section 22 with $m = 8$, 24% of the time with $m = 16$ and 21% of the time with $m = 20$. The larger $m$ is, the more significant the speed-up is.

The loss in performance because of the approximation, on the other hand, is negligible. More specifically, for $r = 8$, performance is decreased by 0.12% for $m = 8$, 0.11% for $m = 16$ and 0.08% for $m = 20$.

## 6    Conclusion

We described an approach to significantly improve the speed of inference with latent-variable PCFGs. The approach approximates tensors which are used in the inside-outside algorithm. The approximation comes with a minimal cost to the performance of the parser. Our algorithm can be used in tandem with estimation algorithms such as EM or spectral algorithms [5]. We note that tensor formulations are used with graphical models [15], for which our technique is also applicable. Similarly, our technique can be applied to other dynamic programming algorithms which compute marginals of a given statistical model.

## Footnotes

[1] All PCFGs in this paper are assumed to be in Chomsky normal form. Our approach generalizes to arbitrary PCFGs, which require tensors of higher rank.

## References

[1] B. W. Bader and T. G. Kolda. Algorithm 862: MATLAB tensor classes for fast algorithm prototyping. *ACM Transactions on Mathematical Software*, 32(4):635–653, 2006.

[2] E. Black, S. Abney, D. Flickenger, C. Gdaniec, R. Grishman, P Harrison, D. Hindle, R. Ingria, F. Jelinek, J. Klavans, M. Liberman, M. Marcus, S. Roukos, B. Santorini, and T. Strzalkowski. A procedure for quantitatively comparing the syntactic coverage of English grammars. In *Proc. of DARPA Workshop on Speech and Natural Language*, 1991.

[3] J. D. Carroll and J. J. Chang. Analysis of individual differences in multidimensional scaling via an N-way generalization of Eckart-Young decomposition. *Psychometrika*, 35:283–319, 1970.

[4] E. Charniak and M. Johnson. Coarse-to-fine $n$-best parsing and maxent discriminative reranking. In *Proceedings of ACL*, 2005.

[5] S. B. Cohen, K. Stratos, M. Collins, D. F. Foster, and L. Ungar. Spectral learning of latent-variable PCFGs. In *Proceedings of ACL*, 2012.

[6] N. M. Faber, R. Bro, and P. Hopke. Recent developments in CANDECOMP/PARAFAC algorithms: a critical review. *Chemometrics and Intelligent Laboratory Systems*, 65(1):119–137, 2003.

[7] J. Goodman. Parsing algorithms and metrics. In *Proceedings of ACL*, 1996.

[8] R. A. Harshman. Foundations of the PARAFAC procedure: Models and conditions for an "explanatory" multi-modal factor analysis. *UCLA working papers in phoentics*, 16:1–84, 1970.

[9] J. Høastad. Tensor rank is NP-complete. *Algorithms*, 11:644–654, 1990.

[10] T. G. Kolda and B. W. Bader. Tensor decompositions and applications. *SIAM Rev.*, 51:455–500, 2009.

[11] J. B. Kruskal. Rank, decomposition, and uniqueness for 3-way and N-way arrays. In R. Coppi and S. Bolasco, editors, *Multiway Data Analysis*, pages 7–18, 1989.

[12] C.D. Manning and H. Schütze. *Foundations of Statistical Natural Language Processing*. MIT Press, 1999.

[13] M. P. Marcus, B. Santorini, and M. A. Marcinkiewicz. Building a large annotated corpus of English: The Penn treebank. *Computational Linguistics*, 19:313–330, 1993.

[14] T. Matsuzaki, Y. Miyao, and J. Tsujii. Probabilistic CFG with latent annotations. In *Proceedings of ACL*, 2005.

[15] A. Parikh, L. Song, and E. P. Xing. A spectral algorithm for latent tree graphical models. In *Proceedings of The 28th International Conference on Machine Learningy (ICML 2011)*, 2011.

[16] S. Petrov, L. Barrett, R. Thibaux, and D. Klein. Learning accurate, compact, and interpretable tree annotation. In *Proceedings of COLING-ACL*, 2006.

